# Structural Risk Minimization for Nonparametric Time Series Prediction

**Ron Meir**[*]
Department of Electrical Engineering
Technion, Haifa 32000, Israel
rmeir@dumbo.technion.ac.il

## Abstract

The problem of time series prediction is studied within the uniform convergence framework of Vapnik and Chervonenkis. The dependence inherent in the temporal structure is incorporated into the analysis, thereby generalizing the available theory for memoryless processes. Finite sample bounds are calculated in terms of covering numbers of the approximating class, and the tradeoff between approximation and estimation is discussed. A complexity regularization approach is outlined, based on Vapnik's method of Structural Risk Minimization, and shown to be applicable in the context of mixing stochastic processes.

## 1 Time Series Prediction and Mixing Processes

A great deal of effort has been expended in recent years on the problem of deriving robust distribution-free error bounds for learning, mainly in the context of memoryless processes (e.g. [9]). On the other hand, an extensive amount of work has been devoted by statisticians and econometricians to the study of parametric (often linear) models of time series, where the dependence inherent in the sample, precludes straightforward application of many of the standard results form the theory of memoryless processes. In this work we propose an extension of the framework pioneered by Vapnik and Chervonenkis to the problem of time series prediction. Some of the more elementary proofs are sketched, while the main technical results will be proved in detail in the full version of the paper.

Consider a stationary stochastic process $\bar{X} = \{\cdots, X_{-1}, X_0, X_1, \ldots\}$, where $X_i$ is a random variable defined over a compact domain in $\mathbf{R}$ and such that $|X_i| \leq B$ with probability 1, for some positive constant $B$. The problem of one-step prediction, in the mean square sense, can then be phrased as that of finding a function $f(\cdot)$ of the infinite past, such that $\mathrm{E}\left|X_0 - f(X_{-\infty}^{-1})\right|^2$ is minimal, where we use the notation $X_i^j = (X_i, X_{i+1}, \ldots, X_j)$,

---

[*]This work was supported in part by the a grant from the Israel Science Foundation

$j \geq i$. It is well known that the optimal predictor in this case is given by the conditional mean, $E[X_0|X_{-\infty}^{-1}]$. While this solution, in principle, settles the issue of optimal predic- tion, it does not settle the issue of actually computing the optimal predictor. First of all, note that to compute the conditional mean, the probabilistic law generating the stochastic process $\bar{X}$ must be known. Furthermore, the requirement of knowing the full past, $X_{-\infty}^{-1}$, is of course rather stringent. In this work we consider the more practical situation, where a *finite* sub-sequence $X_1^N = (X_1, X_2, \cdots, X_N)$ is observed, and an optimal prediction is needed, conditioned on this data. Moreover, for each finite sample size $N$ we allow the predictors to be based only on a finite *lag* vector of size $d$. Ultimately, in order to achieve full generality one may let $d \to \infty$ when $N \to \infty$ in order to obtain the optimal predictor.

We first consider the problem of selecting an empirical estimator from a class of functions $\mathcal{F}_{d,n} : \mathbf{R}^d \to \mathbf{R}$, where $n$ is a complexity index of the class (for example, the number of computational nodes in a feedforward neural network with a single hidden layer), and $|f| \leq B$ for $f \in \mathcal{F}_{d,n}$. Consider then an empirical predictor $f_{d,n,N}(X_{i-d}^{i-1})$, $i > N$, for $X_i$ based on the finite data set $X_1^N$ and depending on the $d$-dimensional lag vector $X_{i-d}^{i-1}$, where $f_{d,n,N} \in \mathcal{F}_{d,n}$. It is possible to split the error incurred by this predictor into three terms, each possessing a rather intuitive meaning. It is the competition between these terms which determines the optimal solution, for a *fixed* amount of data. First, define the loss of a functional predictor $f : \mathbf{R}^d \to \mathbf{R}$ as $L(f) = E\left|X_i - f(X_{i-d}^{i-1})\right|^2$, and let $f_{d,n}^*$ be the optimal function in $\mathcal{F}_{d,n}$ minimizing this loss. Furthermore, denote the optimal lag $d$ predictor by $f_d^*$, and its associated loss by $L_d^*$. We are then able to split the loss of the empirical predictor $f_{d,n,N}$ into three basic components,

$$L(f_{d,n,N}) = \left(L_{d,n,N} - L_{d,n}^*\right) + \left(L_{d,n}^* - L_d^*\right) + L_d^*, \tag{1}$$

where $L_{d,n,N} = L(f_{d,n,N})$. The third term, $L_d^*$, is related to the error incurred in using a fi- nite memory model (of lag size $d$) to predict a process with potentially infinite memory. We do not at present have any useful upper bounds for this term, which is related to the rate of convergence in the martingale convergence theorem, which to the best of our knowledge is unknown for the type of mixing processes we study in this work. The second term in (1), is related to the so-called *approximation error*, given by $E|f_d^*(X_{i-d}^{i-1}) - f_{d,n}^*(X_{i-d}^{i-1})|^2$ to which it can be immediately related through the inequality $||a|^p - |b|^p| \leq p|a - b|| \max(a, b)|^{p-1}$. This term measures the excess error incurred by selecting a function $f$ from a class of lim- ited complexity $\mathcal{F}_{d,n}$, while the optimal lag $d$ predictor $f_d^*$ may be arbitrarily complex. Of course, in order to bound this term we will have to make some regularity assumptions about the latter function. Finally, the first term in (1) represents the so called *estimation error*, and is the only term which depends on the data $X_1^N$. Similarly to the problem of regression for i.i.d. data, we expect that the approximation and estimation terms lead to conflicting demands on the choice of the the complexity, $n$, of the functional class $\mathcal{F}_{d,n}$. Clearly, in order to minimize the approximation error the complexity should be made as large as pos- sible. However, doing this will cause the estimation error to increase, because of the larger freedom in choosing a specific function in $\mathcal{F}_{d,n}$ to fit the data. However, in the case of time series there is an additional complication resulting from the fact that the misspecification error $L_d^*$ is minimized by choosing $d$ to be as large as possible, while this has the effect of increasing both the approximation as well as the estimation errors. We thus expect that some optimal values of $d$ and $n$ exist for each sample size $N$.

Up to this point we have not specified how to select the empirical estimator $f_{d,n,N}$. In this work we follow the ideas of Vapnik [8], which have been studied extensively in the con- text of i.i.d observations, and restrict our selection to that hypothesis which minimizes the empirical error, given by $L_N(f) = \frac{1}{N-d}\sum_{i=d+1}^{N}\left|X_i - f(X_{i-d}^{i-1})\right|^2$. For this function it is easy to establish (see for example [8]) that $(L_{d,n,N} - L_{d,n}^*) \leq 2\sup_{f \in \mathcal{F}_{d,n}}|L(f) - L_N(f)|$. The main distinction from the i.i.d case, of course, is that random variables appearing in

the empirical error, $L_N(f)$, are no longer independent. It is clear at this point that some assumptions are needed regarding the stochastic process $\bar{X}$, in order that a law of large numbers may be established. In any event, it is obvious that the standard approach of using randomization and symmetrization as in the i.i.d case [3] will not work here. To circumvent this problem, two approaches have been proposed. The first makes use of the so-called method of sieves together with extensions of the Bernstein inequality to dependent data [6]. The second approach, to be pursued here, is based on mapping the problem onto one characterized by an i.i.d process [10], and the utilization of the standard results for the latter case.

In order to have some control of the estimation error discussed above, we will restrict ourselves in this work to the class of so-called *mixing* processes. These are processes for which the 'future' depends only weakly on the 'past', in a sense that will now be made precise. Following the definitions and notation of Yu [10], which will be utilized in the sequel, let $\sigma_l = \sigma(X_1^l)$ and $\sigma'_{l+m} = \sigma(X_{l+m}^\infty)$, be the sigma-algebras of events generated by the random variables $X_1^l = (X_1, X_2, \ldots, X_l)$ and $X_{l+m}^\infty = (X_{l+m}, X_{l+m+1}, \ldots)$, respectively. We then define $\beta_m$, the coefficient of absolute regularity, as $\beta_m = \sup_{l \geq 1} \mathrm{E} \sup \left\{ |P(B|\sigma_l) - P(B)| \, : \, B \in \sigma'_{l+m} \right\}$, where the expectation is taken with respect to $\sigma_l = \sigma(X_1^l)$. A stochastic process is said to be $\beta$-*mixing* if $\beta_m \to 0$ as $m \to \infty$. We note that there exist many other definitions of mixing (see [2] for details). The motivation for using the $\beta$-mixing coefficient is that it is the weakest form of mixing for which uniform laws of large numbers can be established. In this work we consider two type of processes for which this coefficient decays to zero, namely *algebraically* decaying processes for which $\beta_m \leq \bar{\beta} m^{-r}, \bar{\beta}, r > 0$, and *exponentially* mixing processes for which $\beta_m \leq \tilde{\beta} \exp\{-bm^\kappa\}, \tilde{\beta}, b, \kappa > 0$. Note that for Markov processes mixing implies exponential mixing, so that at least in this case, there is no loss of generality in assuming that the process is exponentially mixing. Note also that the usual i.i.d process may be obtained from either the exponentially or algebraically mixing process, by taking the limit $\kappa \to \infty$ or $r \to \infty$, respectively.

In this section we follow the approach taken by Yu [10] in deriving uniform laws of large numbers for mixing processes, extending her mainly asymptotic results to finite sample behavior, and somewhat broadening the class of processes considered by her. The basic idea in [10], as in many related approaches, involves the construction of an *independent-block* sequence, which is shown to be 'close' to the original process in a well-defined probabilistic sense. We briefly recapitulate the construction, slightly modifying the notation in [10] to fit in with the present paper. Divide the sequence $X_1^N$ into $2\mu_N$ blocks, each of size $a_N$; we assume for simplicity that $N = 2\mu_N a_N$. The blocks are then numbered according to their order in the block-sequence. For $1 \leq j \leq \mu_N$ define $H_j = \{i : 2(j-1)a_N + 1 \leq i \leq (2j-1)a_N\}$ and $T_j = \{i : (2j-1)a_N + 1 \leq i \leq (2j)a_N\}$. Denote the random variables corresponding to the $H_j$ and $T_j$ indices as $X^{(j)} = \{X_i : i \in H_j\}$ and $X'^{(j)} = \{X_i : i \in T_j\}$. The sequence of H-blocks is then denoted by $X_{a_N} = \{X^{(j)}\}_{j=1}^{\mu_N}$. Now, construct a sequence of independent and identically distributed (i.i.d.) blocks $\{\Xi^{(j)}\}_{j=1}^{\mu_N}$, where $\Xi^{(j)} = \{\xi_i : i \in H_j\}$, such that the sequence is independent of $X_1^N$ and each block has the same distribution as the block $X^{(j)}$ from the original sequence. Because the process is stationary, the blocks $\Xi^{(j)}$ are not only independent but also identically distributed. The basic idea in the construction of the independent block sequence is that it is 'close', in a well-defined sense to the original blocked sequence $X_{a_N}$. Moreover, by appropriately selecting the number of blocks, $\mu_N$, depending on the mixing nature of the sequence, one may relate properties of the original sequence $X_1^N$, to those of the independent block sequence $\Xi_{a_N}$ (see Lemma 4.1 in [10]).

Let $\mathcal{F}$ be a class of bounded functions, such that $0 \leq f \leq B$ for any $f \in \mathcal{F}$. In order to

relate the uniform deviations (with respect to $\mathcal{F}$) of the original sequence $X_1^N$ to those of the independent-block sequence $\Xi_{a_N}$, use is made of Lemma 4.1 from [10]. We also utilize Lemma 4.2 from [10] and modify it so that it holds for finite sample size. Consider the block-independent sequence $\Xi_{a_N}$ and define $\tilde{\mathrm{E}}_{\mu_N} \tilde{f} = \frac{1}{\mu_N} \sum_{j=1}^{\mu_N} \tilde{f}(\Xi^{(j)})$ where $\tilde{f}(\Xi^{(j)}) = \sum_{i \in H_j} f(\xi_i)$, $j = 1, 2, \ldots, \mu_N$, is a sequence of independent random variables such that $|\tilde{f}| \leq a_N B$. In the remainder of the paper we use variables with a tilde above them to denote quantities related to the transformed block sequence. Finally, we use the symbol $\mathrm{E}_N$ to denote the empirical average with respect to the original sequence, namely $\mathrm{E}_N f = (N - d)^{-1} \sum_{i=d+1}^{N} f(X_i)$. The following result can be proved by a simple extension of Lemma 4.2 in [10].

**Lemma 1.1** *Suppose $\mathcal{F}$ is a permissible class of bounded functions, $|f| \leq B$ for $f \in \mathcal{F}$. Then*

$$\mathbf{P}\left\{ \sup_{f \in \mathcal{F}} |\mathrm{E}_N f - \mathrm{E}f| > \varepsilon \right\} \leq 2\tilde{P}\left\{ \sup_{f \in \mathcal{F}} |\tilde{\mathrm{E}}_{\mu_N} \tilde{f} - \tilde{\mathrm{E}}\tilde{f}| > a_N \varepsilon \right\} + 2\mu_N \beta_{a_N}. \quad (2)$$

The main merit of Lemma 1.1 is in the transformation of the problem from the domain of dependent processes, implicit in the quantity $|\mathrm{E}_N f - \mathrm{E}f|$, to one characterized by independent processes, implicit in the term $\tilde{\mathrm{E}}_{\mu_N} \tilde{f} - \tilde{\mathrm{E}}\tilde{f}$, corresponding to the independent blocks. The price paid for this transformation is the extra term $2\mu_N \beta_{a_N}$ which appears on the r.h.s of the inequality appearing in Lemma 1.1.

## 2 Error Bounds

The development in Section 1 was concerned with a scalar stochastic process $\bar{X}$. In order to use the results in the context of time series, we first define a new vector-valued process $\bar{X}' = \{\cdots, \vec{X}_{-1}, \vec{X}_0, \vec{X}_1, \ldots\}$ where $\vec{X}_i = (X_i, X_{i-1}, \ldots, X_{i-d}) \in \mathbf{R}^{d+1}$. For this sequence the $\beta$-mixing coefficients obey the inequality $\beta_m(\bar{X}') \leq \beta_{m-d}(\bar{X})$. Let $\mathcal{F}$ be a space of functions mapping $\mathbf{R}^d \to \mathbf{R}$, and for each $f \in \mathcal{F}$ let the loss function be given by $\ell_f(X_{i-d}^i) = |X_i - f(X_{i-d}^{i-1})|^2$. The loss space is given by $\mathcal{L}_{\mathcal{F}} = \{\ell_f : f \in \mathcal{F}\}$. It is well known in the theory of empirical processes (see [7] for example), that in order to obtain upper bounds on uniform deviations of i.i.d sequences, use must be made of the so-called covering number of the function class $\mathcal{F}$, with respect to the empirical $l_{1,N}$ norm, given by $l_{1,N}(f, g) = N^{-1} \sum_{i=1}^{N} |f(X_i) - g(X_i)|$. Similarly, we denote the empirical norm with respect to the independent block sequence by $\tilde{l}_{1,\mu_N}$, where $\tilde{l}_{1,\mu_N}(f, g) = \mu_N^{-1} \sum_{j=1}^{\mu_N} |\tilde{f}(X^{(j)}) - \tilde{g}(X^{(j)})|$, and where $\tilde{f}(X^{(j)}) = \sum_{i \in H_j} X_i$ and similarly for $\tilde{g}$. Following common practice we denote the $\varepsilon$-covering number of the functional space $\mathcal{F}$ using the metric $\rho$ by $\mathcal{N}(\varepsilon, \mathcal{F}, \rho)$.

**Definition 1** *Let $\mathcal{L}_{\mathcal{F}}$ be a class of real-valued functions from $\mathbf{R}^D \to \mathbf{R}$, $D = d + 1$. For each $\ell_f \in \mathcal{L}_{\mathcal{F}}$ and $\mathbf{x} = (\vec{x}_1, \vec{x}_2, \ldots, \vec{x}_{a_N})$, $\vec{x}_i \in \mathbf{R}^D$, let $\tilde{\ell}_f(\mathbf{x}) = \sum_{i=1}^{a_N} \ell_f(\vec{x}_i)$. Then define $\tilde{\mathcal{L}}_{\mathcal{F}} = \left\{ \tilde{\ell}_f : \ell_f \in \mathcal{L}_{\mathcal{F}} \right\}$, where $\tilde{\ell}_f : \mathbf{R}^{a_N D} \to \mathbf{R}^+$.*

In order to obtain results in terms of the covering numbers of the space $\mathcal{L}_{\mathcal{F}}$ rather than $\tilde{\mathcal{L}}_{\mathcal{F}}$, which corresponds to the transformed sequence, we need the following lemma, which is not hard to prove.

**Lemma 2.1** *For any $\varepsilon > 0$*

$$\mathcal{N}\left(\varepsilon, \tilde{\mathcal{L}}_{\mathcal{F}}, \tilde{l}_{1,\mu_N}\right) \leq \mathcal{N}\left(\varepsilon/a_N, \mathcal{L}_{\mathcal{F}}, l_{1,N}\right).$$

PROOF The result follows by sequence of simple inequalities, showing that $\tilde{l}_{1,\mu_N}(\tilde{f}, \tilde{g}) \leq a_N l_{1,N}(f, g)$. ∎

We now present the main result of this section, namely an upper bound for the uniform deviations of mixing processes, which in turn yield upper bounds on the error incurred by the empirically optimal predictor $f_{d,n,N}$.

**Theorem 2.1** *Let $\bar{X} = \{\dots, X_1, X_0, X_1, \dots\}$ be a bounded stationary $\beta$-mixing stochastic process, with $|X_i| \leq B$, and let $\mathcal{F}$ be a class of bounded functions, $f : \mathbf{R}^d \rightharpoonup [0, B]$. For each sample size $N$, let $\hat{f}_N$ be the function in $\mathcal{F}$ which minimizes the empirical error, and $f^*$ is the function in $\mathcal{F}$ minimizing the true error $L(f)$. Then,*

$$\mathbf{P}\left\{L(\hat{f}_N) - L(f^*) > \varepsilon\right\} \leq 8E\mathcal{N}(\varepsilon', \mathcal{F}, l_{1,N}) \exp\left\{-\frac{\mu_N \varepsilon^2}{64(2B)^4}\right\} + 2\mu_N \beta_{a_N - d}. \tag{3}$$

*where $\varepsilon' = \varepsilon/128B$.*

PROOF The theorem is established by making use of Lemma 1.1, and the basic results from the theory of uniform convergence for i.i.d. processes, together with Lemma 2.1 relating the covering numbers of the spaces $\tilde{\mathcal{L}}_{\mathcal{F}}$ and $\mathcal{L}_{\mathcal{F}}$. The covering numbers of $\mathcal{L}_{\mathcal{F}}$ and $\mathcal{F}$ are easily related using $\mathcal{N}(\varepsilon, \mathcal{L}_{\mathcal{F}}, L_1(P)) \leq \mathcal{N}(\varepsilon/2B, \mathcal{F}, L_1(P))$. ∎

Up to this point we have not specified $\mu_N$ and $a_N$, and the result is therefore quite general. In order to obtain weak consistency we require that that the r.h.s. of (3) converge to zero for each $\varepsilon > 0$. This immediately yields the following conditions on $\mu_N$ (and thus also on $a_N$ through the condition $2a_N\mu_N = N$).

**Corollary 2.1** *Under the conditions of Theorem 2.1, and the added requirements that $d = o(a_N)$ and $\mathcal{N}(\varepsilon, \mathcal{F}, l_{1,N}) < \infty$, the following choices of $\mu_N$ are sufficient to guarantee the weak consistency of the empirical predictor $\hat{f}_N$:*

$$\mu_N \sim N^{\kappa/(1+\kappa)} \qquad \text{(exponential mixing)}, \tag{4}$$

$$\mu_N \sim N^{s/(1+s)}, \, 0 < s < r \qquad \text{(algebraic mixing)}, \tag{5}$$

*where the notation $a_N \sim b_N$ implies that $\Omega(b_N) \leq a_N \leq O(b_N)$.*

PROOF Consider first the case of exponential mixing. In this case the r.h.s. of (3) clearly converges to zero because of the finiteness of the covering number. The fastest rate of convergence is achieved by balancing the two terms in the equation, leading to the choice $\mu_N \sim N^{\kappa/(1+\kappa)}$. In the case of algebraic mixing, the second term on the r.h.s. of (3) is of the order $O(\mu_N a_N^{-r})$ where we have used $d = o(a_N)$. Since $\mu_N a_N \sim N$, a sufficient condition to guarantee that this term converge to zero is that $\mu_N \sim N^{s/(1+s)}$, $0 < s < r$, as was claimed. ∎

In order to derive bounds on the expected error, we need to make an assumption concerning the covering number of the space $\mathcal{F}$. In particular, we know from the work Haussler [4] that the covering number is upper bounded as follows

$$\mathcal{N}(\varepsilon, \mathcal{F}, L_1(P)) \leq e(\text{Pdim}(\mathcal{F}) + 1)\left(\frac{2eB}{\varepsilon}\right)^{\text{Pdim}(\mathcal{F})},$$

for any measure $P$. Thus, assuming the finiteness of the pseudo-dimension of $\mathcal{F}$ guarantees a finite covering number.

## 3   Structural Risk Minimization

The results in Section 2 provide error bounds for estimators formed by minimizing the empirical error over a fixed class of $d$-dimensional functions. It is clear that the complexity of the class of functions plays a crucial role in the procedure. If the class is too rich, manifested by very large covering numbers, clearly the estimation error term will be very large. On the other hand, biasing the class of functions by restricting its complexity, leads to poor approximation rates. A well-known strategy for overcoming this dilemma is obtained by considering a hierarchy of functional classes with increasing complexity. For any given sample size, the optimal trade-off between estimation and approximation can then be determined by balancing the two terms. Such a procedure was developed in the late seventies by Vapnik [8], and termed by him *structural risk minimization* (SRM). Other more recent approaches, collectively termed complexity regularization, have been extensively studied in recent years (e.g. [1]). It should be borne in mind, however, that in the context of time series there is an added complexity, that does not exist in the case of regression. Recall that the results derived in Section 2 assumed some fixed lag vector $d$. In general the optimal value of $d$ is unknown, and could in fact be infinite. In order to achieve optimal performance in a nonparametric setting, it is crucial that the size of the lag be chosen adaptively as well. This added complexity needs to be incorporated into the SRM framework, if optimal performance in the face of unknown memory size is to be achieved.

Let $\mathcal{F}_{d,n}$, $d, n \in \mathbf{N}$ be a sequence of functions, and define $\mathcal{F} = \bigcup_{d=1}^{\infty} \bigcup_{n=1}^{\infty} \mathcal{F}_{d,n}$. For any $\mathcal{F}_{d,n}$ let

$$\mathcal{N}_1(\varepsilon, \mathcal{F}_{d,n}) = \sup_{x^N} \mathcal{N}(\varepsilon, \mathcal{F}_{d,n}, l_{1,N}),$$

which from [4] is upper bounded by $c\varepsilon^{-\mathrm{Pdim}(\mathcal{F}_{d,n})}$. We observe in passing that Lugosi and Nobel [5] have recently considered situations where the pseudo-dimension $\mathrm{Pdim}(\mathcal{F}_{d,n})$ is unknown, and the covering number is estimated empirically from the data. Although this line of thought is potentially very useful, we do not pursue it here, but rather assume that upper bounds on the pseudo-dimensions of $\mathcal{F}_{d,n}$ are known, as is the case for many classes of functions used in practice (see for example [9]).

In line with the standard approach in [8] we introduce a new empirical function, which takes into account both the empirical error as well as the complexity costs penalizing overly complex models (large complexity index $n$ and lag size $d$). Let

$$\tilde{L}_{d,n,N}(f) = L_N(f) + \Delta_{d,n,N}(\varepsilon) + \Delta_{d,N}, \tag{6}$$

where $L_N(f)$ is the empirical error of the predictor $f$ and the complexity penalties $\Delta$ are given by

$$\Delta_{d,n,N}(\varepsilon) = \sqrt{\frac{\log \mathcal{N}_1(\varepsilon, \mathcal{F}_{d,n}) + c_n}{\mu_N/64(2B)^4}} \tag{7}$$

$$\Delta_{d,N} = \sqrt{\frac{c_d}{\mu_N/64(2B)^4}}. \tag{8}$$

The specific form and constants in these definitions are chosen with hindsight, so as to achieve the optimal rates of convergence in Theorem 3.1 below. The constants $c_n$ and $c_d$ are positive constants obeying $\sum_{n=1}^{\infty} e^{-c_n} \leq 1$ and similarly for $c_d$. A possible choice is $c_n = 2 \log n + 1$ and $c_d = 2 \log d + 1$. The value of $\mu_N$ can be chosen in accordance with Corollary 2.1.

Let $\hat{f}_{d,n,N}$ minimize the empirical error $L_N(f)$ within the class of functions $\mathcal{F}_{d,n}$. We assume that the classes $\mathcal{F}_{d,n}$ are compact, so that such a minimizer exists. Further, let $\hat{f}_N$

be the function in $\mathcal{F}$ minimizing the complexity penalized loss (6), namely

$$\tilde{L}_{d,n,N}(\hat{f}_N) = \min_{d \geq 1} \min_{n \geq 1} \tilde{L}_{d,n,N}(\hat{f}_{d,n,N}) \tag{9}$$

The following basic result establishes the consistency of the structural risk minimization approach, and yields upper bounds on its performance.

**Theorem 3.1** *Let $\mathcal{F}_{d,n}$, $d, n \in \mathbf{N}$ be sequence of functional classes, where $f \in \mathcal{F}_{d,n}$ is a mapping from $\mathbf{R}^d$ to $\mathbf{R}$. The expected loss of the function $\hat{f}_N$, selected according to the SRM principle, is upper bounded by*

$$\mathrm{E}L(\hat{f}_N) \leq \min_{d,n} \left\{ \inf_{d,n} L(f) + c_1 \sqrt{\frac{\gamma_{d,n} \log \mu_N + c_n}{\mu_N}} + \sqrt{\frac{c_2}{\mu_N}} \right\} + 4(2B)^2 \mu_N \beta_{a_N/2}. \tag{10}$$

The main merit of Theorem 3.1 is the demonstration that the SRM procedure achieves an optimal balance between approximation and estimation, while retaining its nonparametric attributes. In particular, if the optimal lag $d$ predictor $f_d^*$ belongs to $\mathcal{F}_{d,n_0}$ for some $n_0$, the SRM predictor would converge to it at the same rate as if $n_0$ were known in advance. The same type of adaptivity is obtained with respect to the lag size $d$. The nonparametric rates of convergence of the SRM predictor will be discussed in the full paper.

# References

[1] A. Barron. Complexity Regularization with Application to Artificial Neural Networks. In G. Roussas, editor, *Nonparametric Functional Estimation and Related Topics*, pages 561–576. Kluwer Academic Press, 1991.

[2] L. Györfi, W. Härdle, P. Sarda, and P. Vieu. *Nonparametric Curve Estimation from Time Series*. Springer Verlag, New York, 1989.

[3] D. Haussler. Decision Theoretic Generalizations of the PAC Model for Neural Net and Other Learning Applications. *Information and Computation*, 100:78–150, 1992.

[4] D. Haussler. Sphere Packing Numbers for Subsets of the Boolean $n$-Cube with Bounded Vapnik-Chervonenkis Dimesnion. *J. Combinatorial Theory*, Series A 69:217–232, 1995.

[5] G. Lugosi and A. Nobel. Adaptive Model Selection Using Empirical Complexities. Submitted to *Annals Statis.*, 1996.

[6] D . Modha and E. Masry. Memory Universal Prediction of Stationary Random Processes. *IEEE Trans. Inf. Th.*, January, 1998.

[7] D. Pollard. *Convergence of Empirical Processes*. Springer Verlag, New York, 1984.

[8] V. N. Vapnik. *Estimation of Dependences Based on Empirical Data*. Springer Verlag, New York, 1992.

[9] M. Vidyasagar. *A Theory of Learning and Generalization*. Springer Verlag, New York, 1996.

[10] B. Yu. Rates of convergence for empirical processes of stationary mixing sequences. *Annals of Probability*, 22:94–116, 1984.
